# A mean field algorithm for Bayes learning in large feed-forward neural networks

**Manfred Opper**
Institut für Theoretische Physik
Julius-Maximilians-Universität, Am Hubland
D-97074 Würzburg, Germany
opper@physik.Uni-Wuerzburg.de

**Ole Winther**
CONNECT
The Niels Bohr Institute
Blegdamsvej 17
2100 Copenhagen, Denmark
winther@connect.nbi.dk

## Abstract

We present an algorithm which is expected to realise Bayes optimal predictions in large feed-forward networks. It is based on mean field methods developed within statistical mechanics of disordered systems. We give a derivation for the single layer perceptron and show that the algorithm also provides a leave-one-out cross-validation test of the predictions. Simulations show excellent agreement with theoretical results of statistical mechanics.

## 1 INTRODUCTION

Bayes methods have become popular as a consistent framework for regularization and model selection in the field of neural networks (see e.g. [MacKay,1992]). In the Bayes approach to statistical inference [Berger,1985] one assumes that the prior uncertainty about parameters of an unknown data generating mechanism can be encoded in a probability distribution, the so called *prior*. Using the prior and the likelihood of the data given the parameters, the *posterior* distribution of the parameters can be derived from Bayes rule. From this posterior, various estimates for functions of the parameter, like predictions about unseen data, can be calculated. However, in general, those predictions cannot be realised by specific parameter values, but only by an ensemble average over parameters according to the posterior probability.

Hence, exact implementations of Bayes method for neural networks require averages over network parameters which in general can be performed by time consuming

Monte Carlo procedures. There are however useful approximate approaches for calculating posterior averages which are based on the assumption of a Gaussian form of the posterior distribution [MacKay,1992]. Under regularity conditions on the likelihood, this approximation becomes asymptotically exact when the number of data is large compared to the number of parameters. This Gaussian ansatz for the posterior may not be justified when the number of examples is small or comparable to the number of network weights. A second cause for its failure would be a situation where discrete classification labels are produced from a probability distribution which is a nonsmooth function of the parameters. This would include the case of a network with *threshold* units learning a *noise free* binary classification problem.

In this contribution we present an alternative approximate realization of Bayes method for neural networks, which is not based on asymptotic posterior normality. The posterior averages are performed using mean field techniques known from the statistical mechanics of disordered systems. Those are expected to become exact in the limit of a large number of network parameters under additional assumptions on the statistics of the input data. Our analysis follows the approach of [Thouless, Anderson& Palmer,1977] (TAP) as adapted to the simple perceptron by [Mézard,1989].

The basic set up of the Bayes method is as follows: We have a training set consisting of $m$ input-output pairs $D_m = \{(\mathbf{s}^\mu, \sigma^\mu), m = 1, \ldots, \mu\}$, where the outputs are generated independently from a conditional probability distribution $P(\sigma^\mu|\mathbf{w}, \mathbf{s}^\mu)$. This probability is assumed to describe the output $\sigma^\mu$ to an input $\mathbf{s}^\mu$ of a neural network with weights $\mathbf{w}$ subject to a suitable noise process. If we assume that the unknown parameters $\mathbf{w}$ are randomly distributed with a prior distribution $p(\mathbf{w})$, then according to Bayes theorem our knowledge about $\mathbf{w}$ after seeing $m$ examples is expressed through the posterior distribution

$$p(\mathbf{w}|D_m) = Z^{-1}p(\mathbf{w}) \prod_{\mu=1}^{m} P(\sigma^\mu|\mathbf{w}, \mathbf{s}^\mu) \tag{1}$$

where $Z = \int d\mathbf{w} p(\mathbf{w}) \prod_{\mu=1}^{m} P(\sigma^\mu|\mathbf{w}, \mathbf{s}^\mu)$ is called the partition function in statistical mechanics and the *evidence* in Bayesian terminology. Taking the average with respect to the posterior eq. (1), which in the following will be denoted by angle brackets, gives Bayes estimates for various quantities. For example the optimal predictive probability for an output $\sigma$ to a new input $\mathbf{s}$ is given by $\hat{P}^{\text{Bayes}}(\sigma|\mathbf{s}) = \langle P(\sigma|\mathbf{w}, \mathbf{s}) \rangle$.

In section 2 exact equations for the posterior averaged weights $\langle \mathbf{w} \rangle$ are derived for arbitrary networks. In 3 we specialize these equations to a perceptron and develop a mean field ansatz in section 4. The resulting system of mean field equations equations is presented in section 5. In section 6 we consider Bayes optimal predictions and a leave-one-out estimator for the generalization error. We conclude in section 7 with a discussion of our results.

## 2   A RESULT FOR POSTERIOR AVERAGES FROM GAUSSIAN PRIORS

In this section we will derive an interesting equation for the posterior mean of the weights for arbitrary networks when the prior is Gaussian. This average of the

weights can be calculated for the distribution (1) by using the following simple and well known result for averages over Gaussian distributions.

Let $v$ be a Gaussian random variable with zero means. Then for any function $f(v)$, we have

$$\langle v f(v) \rangle_G = \langle v^2 \rangle_G \cdot \langle \frac{df(v)}{dv} \rangle_G. \qquad (2)$$

Here $\langle \ldots \rangle_G$ denotes the average over the Gaussian distribution of $v$. The relation is easily proved from an integration by parts.

In the following we will specialize to an isotropic Gaussian prior $p(\mathbf{w}) = \frac{1}{\sqrt{2\pi}^N} e^{-\frac{1}{2}\mathbf{w}\cdot\mathbf{w}}$. In [Opper & Winter,1996] anisotropic priors are treated as well. Applying (2) to each component of $\mathbf{w}$ and the function $\prod_{\mu=1}^{m} P(\sigma^\mu|\mathbf{w},\mathbf{s}^\mu)$, we get the following equations

$$\langle \mathbf{w} \rangle = Z^{-1} \int d\mathbf{w} \ \mathbf{w} p(\mathbf{w}) \prod_{\nu=1}^{m} P(\sigma^\nu|\mathbf{w},\mathbf{s}^\nu)$$

$$= Z^{-1} \sum_{\mu=1}^{m} \int d\mathbf{w} p(\mathbf{w}) \prod_{\nu\neq\mu}^{m} P(\sigma^\nu|\mathbf{w},\mathbf{s}^\nu) \nabla_\mathbf{w} P(\sigma^\mu|\mathbf{w},\mathbf{s}^\mu) \qquad (3)$$

$$= \sum_{\mu=1}^{m} \frac{\langle \nabla_\mathbf{w} P(\sigma^\mu|\mathbf{w},\mathbf{s}^\mu) \rangle_\mu}{\langle P(\sigma^\mu|\mathbf{w},\mathbf{s}^\mu) \rangle_\mu}.$$

Here $\langle \ldots \rangle_\mu = \frac{\int d\mathbf{w} p(\mathbf{w}) \ldots \prod_{\nu\neq\mu} P(\sigma^\nu|\mathbf{w},\mathbf{s}^\nu)}{\int d\mathbf{w} p(\mathbf{w}) \prod_{\nu\neq\mu} P(\sigma^\nu|\mathbf{w},\mathbf{s}^\nu)}$ is a *reduced* average over a posterior where the $\mu$-th example is kept out of the training set and $\nabla_\mathbf{w}$ denotes the gradient with respect to $\mathbf{w}$.

## 3   THE PERCEPTRON

In the following, we will utilize the fact that for neural networks, the probability (1) depends only on the so called internal fields $\Delta = \frac{1}{\sqrt{N}}\mathbf{w}\cdot\mathbf{s}$.

A simple but nontrivial example is the perceptron with $N$ dimensional input vector $\mathbf{s}$ and output $\sigma(\mathbf{w},\mathbf{s}) = \text{sign}(\Delta)$. We will generalize the noise free model by considering label noise in which the output is flipped, i.e. $\sigma\Delta < 0$ with a probability $(1+e^\beta)^{-1}$. (For simplicity, we will assume that $\beta$ is known such that no prior on $\beta$ is needed.) The conditional probability may thus be written as

$$P(\sigma^\mu\Delta^\mu) \equiv P(\sigma^\mu|\mathbf{w},\mathbf{s}^\mu) = \frac{e^{-\beta\Theta(-\sigma^\mu\Delta^\mu)}}{1+e^{-\beta}}, \qquad (4)$$

where $\Theta(x) = 1$ for $x > 0$ and 0 otherwise. Obviously, this a nonsmooth function of the weights $\mathbf{w}$, for which the posterior will not become Gaussian asymptotically.

For this case (3) reads

$$\langle \mathbf{w} \rangle = \frac{1}{\sqrt{N}} \sum_{\mu=1}^{m} \frac{\langle P'(\sigma^\mu\Delta^\mu) \rangle_\mu}{\langle P(\sigma^\mu\Delta^\mu) \rangle_\mu} \sigma^\mu\mathbf{s}^\mu = \qquad (5)$$

$$\frac{1}{\sqrt{N}} \sum_{\mu=1}^{m} \frac{\int d\Delta f_\mu(\Delta) P'(\sigma^\mu\Delta)}{\int d\Delta f_\mu(\Delta) P(\sigma^\mu\Delta)} \sigma^\mu\mathbf{s}^\mu$$

$f_\mu(\Delta)$ is the density of $\frac{1}{\sqrt{N}}\mathbf{w} \cdot \mathbf{s}^\mu$, when the weights $\mathbf{w}$ are randomly drawn from a posterior, where example $(\mathbf{s}^\mu, \sigma^\mu)$ was kept out of the training set. This result states that the weights are linear combinations of the input vectors. It gives an example of the ability of Bayes method to regularize a network model: the effective number of parameters will never exceed the number of data points.

## 4   MEAN FIELD APPROXIMATION

Sofar, no approximations have been made to obtain eqs. (3,5). In general $f_\mu(\Delta)$ depends on the entire set of data $D_m$ and can not be calculated easily. Hence, we look for a useful approximation to these densities.

We split the internal field into its average and fluctuating parts, i.e. we set $\Delta^\mu = \langle\Delta^\mu\rangle_\mu + v^\mu$, with $v^\mu = \frac{1}{\sqrt{N}}(\mathbf{w} - \langle\mathbf{w}\rangle_\mu)\mathbf{s}^\mu$. Our mean field approximation is based on the assumption of a central limit theorem for the fluctuating part of the internal field, $v^\mu$ which enters in the reduced average of eq. (5). This means, we assume that the *non-Gaussian fluctuations* of $w_i$ around $\langle w_i\rangle_\mu$, when mulitplied by $s_i^\mu$ will sum up to make $v^\mu$ a Gaussian random variable. The important point is here that for the reduced average, the $w_i$ are not correlated to the $s_i^\mu$! [1]

We expect that this Gaussian approximation is reasonable, when $N$, the number of network weights is sufficiently large. Following ideas of [Mézard, Parisi & Virasoro,1987] and [Mézard,1989], who obtained mean field equations for a variety of disordered systems in statistical mechanics, one can argue that in many cases this assumption may be exactly fulfilled in the 'thermodynamic limit' $m, N \to \infty$ with $\alpha = \frac{m}{N}$ fixed. According to this ansatz, we get

$$f_\mu(\Delta) \simeq \frac{\exp\left[-\frac{(\Delta - \langle\Delta^\mu\rangle_\mu)^2}{2\lambda^\mu}\right]}{\sqrt{2\pi\lambda^\mu}}$$

in terms of the second moment of $v^\mu$ $\lambda^\mu \equiv \frac{1}{N}\sum_{i,j} s_i^\mu s_j^\mu (\langle w_i w_j\rangle_\mu - \langle w_i\rangle_\mu \langle w_j\rangle_\mu)$.

To evaluate (5) we need to calculate the mean $\langle\Delta^\mu\rangle_\mu$ and the variance $\lambda^\mu$. The first problem is treated easily within the Gaussian approximation.

$$
\begin{aligned}
\langle\Delta_k^\mu\rangle_\mu &= \langle\Delta^\mu\rangle - \langle v^\mu\rangle \\
&= \langle\Delta^\mu\rangle - \frac{\langle v^\mu P(\sigma^\mu \Delta^\mu)\rangle_\mu}{\langle P(\sigma^\mu \Delta^\mu)\rangle_\mu} \\
&= \langle\Delta^\mu\rangle - \lambda^\mu \frac{\langle P'(\sigma^\mu \Delta^\mu)\rangle_\mu}{\langle P(\sigma^\mu \Delta^\mu)\rangle_\mu} \sigma^\mu
\end{aligned}
\tag{6}
$$

In the third line (2) has been used again for the Gaussian random variable $v^\mu$.

Sofar, the calculation of the variance $\lambda^\mu$ for *general inputs* is an open problem. However, we can make a further reasonable ansatz, when the distribution of the inputs is known. The following approximation for $\lambda^\mu$ is expected to become exact in the thermodynamic limit if the inputs of the training set are drawn independently

from a distribution, where all components $s_i$ are uncorrelated and normalized i.e. $\overline{s_i} = 0$ and $\overline{s_i s_j} = \delta_{ij}$. The bars denote expectation over the distribution of inputs. For the generalisation to a correlated input distribution see [Opper& Winther,1996]. Our basic mean field assumption is that the fluctuations of the $\lambda^\mu$ with the data set can be neglected so that we can replace them by their averages $\overline{\lambda^\mu}$. Since the reduced posterior averages are not correlated with the data $s_i^\mu$, we obtain $\lambda^\mu \simeq \frac{1}{N} \sum_i (\langle w_i^2 \rangle_\mu - \langle w_i \rangle_\mu^2)$. Finally, we replace the reduced average by the expectation over the full posterior, neglecting terms of order $1/N$. Using $\sum_i \langle w_i^2 \rangle = N$, which follows from our choice of the Gaussian prior, we get $\lambda^\mu \simeq \lambda = 1 - \frac{1}{N} \sum_i \langle w_i \rangle^2$. This depends only on known quantities.

## 5 MEAN FIELD EQUATIONS FOR THE PERCEPTRON

(5) and (6) give a selfconsistent set of equations for the variable $x^\mu \equiv \frac{\langle P'(\sigma^\mu \Delta^\mu) \rangle_\mu}{\langle P(\sigma^\mu \Delta^\mu) \rangle_\mu}$ . We finally get

$$\langle \mathbf{w} \rangle = \frac{1}{\sqrt{N}} \sum_{\mu=1}^{m} x^\mu \, \sigma^\mu \mathbf{s}^\mu \tag{7}$$

$$x^\mu = \frac{(1 - e^{-\beta}) e^{-z^{\mu 2}/2}}{\sqrt{2\pi\lambda}[e^{-\beta} + (1 - e^{-\beta}) H(-\sigma^\mu z^\mu)]} \tag{8}$$

with

$$z^\mu = \frac{\frac{1}{\sqrt{N}} \langle \mathbf{w} \rangle \cdot \mathbf{s}^\mu - \lambda \sigma^\mu x^\mu}{\sqrt{\lambda}}. \tag{9}$$

$$H(t) = \int_t^\infty dx e^{-x^2/2} / \sqrt{2\pi}$$

$$\lambda = 1 - \frac{1}{N} \sum_i \langle w_i \rangle^2.$$

These mean field equations can be solved by iteration. It is useful to start with a small number of data and then to increase the number of data in steps of 1 - 10. Numerical work show that the algorithm works well even for small systems sizes, $N \simeq 15$.

## 6 BAYES PREDICTIONS AND LEAVE-ONE-OUT

After solving the mean field equations we can make optimal Bayesian classifications for new data s by chosing the output label with the largest predictive probability. In case of output noise this reduces to $\sigma^{\text{Bayes}}(\mathbf{s}) = \text{sign}\langle \sigma(\mathbf{w}, \mathbf{s}) \rangle$ Since the posterior distribution is independent of the new input vector we can apply the Gaussian assumption again to the internal field, $\Delta$. and obtain $\sigma^{\text{Bayes}}(\mathbf{s}) = \sigma(\langle \mathbf{w} \rangle, \mathbf{s})$, i.e for the simple perceptron the averaged weights implement the Bayesian prediction. This will not be the case for multi-layer neural networks.

We can also get an estimate for the generalization error which occurs on the prediction of new data. The generalization error for the Bayes prediction is defined by $\epsilon^{\text{Bayes}} = \langle \Theta(-\sigma(\mathbf{s}) \langle \sigma(\mathbf{w}, \mathbf{s}) \rangle) \rangle_\mathbf{s}$, where $\sigma(\mathbf{s})$ is the true output and $\langle \ldots \rangle_\mathbf{s}$ denotes average over the input distribution. To obtain the *leave-one-out estimator* of $\epsilon$ one

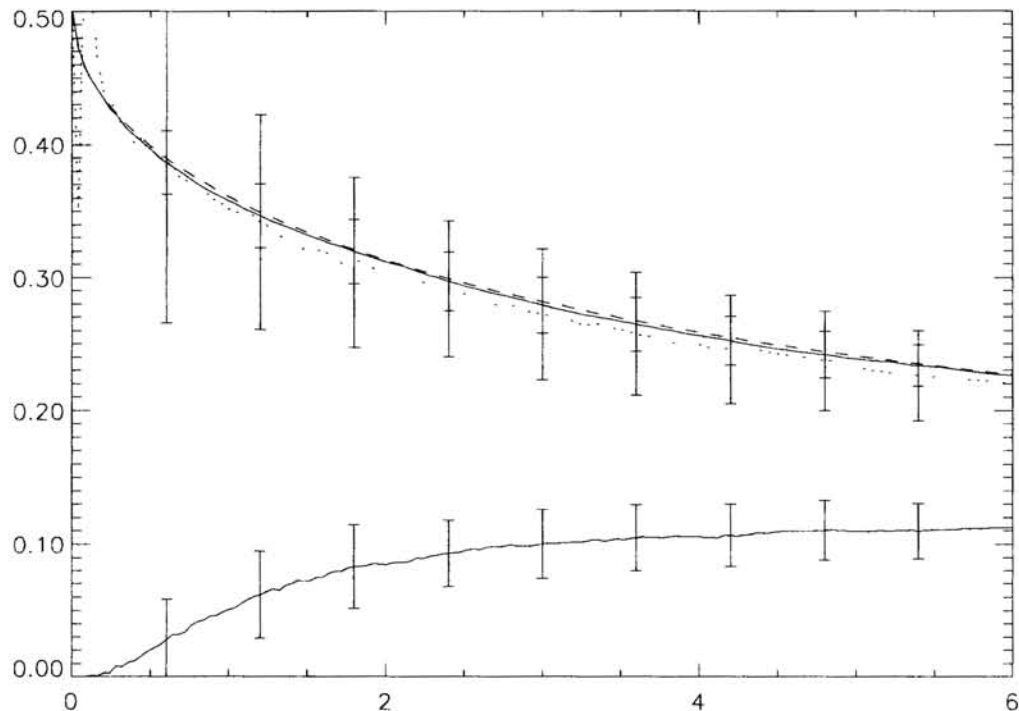

Figure 1: Error vs. $\alpha = m/N$ for the simple perceptron with output noise $\beta = 0.5$ and $N = 50$ averaged over 200 runs. The full lines are the simulation results (upper curve shows prediction error and the lower curve shows training error). The dashed line is the theoretical result for $N \to \infty$ obtained from statistical mechanics [Opper & Haussler,1991]. The dotted line with larger error bars is the moving control estimate.

removes the $\mu$-th example from the training set and trains the network using only the remaining $m - 1$ examples. The $\mu$'th example is used for testing. Repeating this procedure for all $\mu$ an unbiased estimate for the Bayes generalization error with $m-1$ training data is obtained as the mean value $\epsilon_{MC}^{Bayes} = \frac{1}{m} \sum_{\mu} \Theta \left( -\sigma^{\mu} \langle \sigma(\mathbf{w}, \mathbf{s}^{\mu}) \rangle_{\mu} \right)$ which is exactly the type of reduced averages which are calculated within our approach. Figure 1 shows a result of simulations of our algorithm when the inputs are uncorrelated and the outputs are generated from a teacher perceptron with fixed noise rate $\beta$.

## 7   CONCLUSION

In this paper we have presented a mean field algorithm which is expected to implement a Bayesian optimal classification well in the limit of large networks. We have explained the method for the single layer perceptron. An extension to a simple multilayer network, the so called committee machine with a tree architecture is discussed in [Opper& Winther,1996]. The algorithm is based on a Gaussian assumption for the distribution of the internal fields, which seems reasonable for large networks. The main problem sofar is the restriction to ideal situations such as a known distri-

bution of inputs which is not a realistic assumption for real world data. However, this assumption only entered in the calculation of the variance of the Gaussian field. More theoretical work is necessary to find an approximation to the variance which is valid in more general cases. A promising approach is a derivation of the mean field equations directly from an approximation to the free energy $-\ln(Z)$. Besides a deeper understanding this would also give us the possibility to use the method with the so called evidence framework, where the partition function (evidence) can be used to estimate unknown (hyper-) parameters of the model class [Berger,1985]. It will further be important to extend the algorithm to fully connected architectures. In that case it might be necessary to make further approximations in the mean field method.

## ACKNOWLEDGMENTS

This research is supported by a Heisenberg fellowship of the *Deutsche Forschungsgemeinschaft* and by the Danish Research Councils for the Natural and Technical Sciences through the Danish Computational Neural Network Center (CONNECT).

## Footnotes

[1] Note that the fluctuations of the internal field with respect to the *full* posterior mean (which depends on the input $\mathbf{s}^\mu$) is *non* Gaussian, because the different terms in the sum become slightly correlated.

## REFERENCES

Berger, J. O. (1985) *Statistical Decision theory and Bayesian Analysis*, Springer-Verlag, New York.

MacKay, D. J. (1992) *A practical Bayesian framework for backpropagation networks*, Neural Comp. 4 448.

Mézard, M., Parisi G. & Virasoro M. A. (1987) *Spin Glass Theory and Beyond*, Lecture Notes in Physics, 9, World Scientific, .

Mézard, M. (1989) *The space of interactions in neural networks: Gardner's calculation with the cavity method* J. Phys. A **22**, 2181 .

Opper, M. & Haussler, D. (1991) in *IVth Annual Workshop on Computational Learning Theory (COLT91)*, Morgan Kaufmann.

Opper M. & Winther O (1996) *A mean field approach to Bayes learning in feedforward neural networks*, Phys. Rev. Lett. **76** 1964.

Thouless, D.J., Anderson, P. W. & Palmer, R.G. (1977), *Solution of 'Solvable model of a spin glass'* Phil. Mag. **35**, 593.